# Improved Moves for Truncated Convex Models

**M. Pawan Kumar**
**Dept. of Engineering Science**
**University of Oxford**
**pawan@robots.ox.ac.uk**

**P.H.S. Torr**
**Dept. of Computing**
**Oxford Brookes University**
**philiptorr@brookes.ac.uk**

## Abstract

We consider the problem of obtaining the approximate maximum *a posteriori* estimate of a discrete random field characterized by pairwise potentials that form a truncated convex model. For this problem, we propose an improved st-MINCUT based *move making* algorithm. Unlike previous move making approaches, which either provide a loose bound or no bound on the quality of the solution (in terms of the corresponding Gibbs energy), our algorithm achieves the same guarantees as the standard linear programming (LP) relaxation. Compared to previous approaches based on the LP relaxation, e.g. interior-point algorithms or tree-reweighted message passing (TRW), our method is faster as it uses only the efficient st-MINCUT algorithm in its design. Furthermore, it directly provides us with a primal solution (unlike TRW and other related methods which solve the dual of the LP). We demonstrate the effectiveness of the proposed approach on both synthetic and standard real data problems.

Our analysis also opens up an interesting question regarding the relationship between move making algorithms (such as $\alpha$-expansion and the algorithms presented in this paper) and the randomized rounding schemes used with convex relaxations. We believe that further explorations in this direction would help design efficient algorithms for more complex relaxations.

## 1   Introduction

Discrete random fields are a powerful tool for formulating several problems in Computer Vision such as stereo reconstruction, segmentation, image stitching and image denoising [22]. Given data $\mathbf{D}$ (e.g. an image or a video), random fields model the probability of a set of random variables $\mathbf{v}$, i.e. either the joint distribution of $\mathbf{v}$ and $\mathbf{D}$ as in the case of Markov random fields (MRF) [2] or the conditional distribution of $\mathbf{v}$ given $\mathbf{D}$ as in the case of conditional random fields (CRF) [18]. The word 'discrete' refers to the fact that each of the random variables $v_a \in \mathbf{v} = \{v_0, \cdots, v_{n-1}\}$ can take one label from a discrete set $\mathbf{l} = \{l_0, \cdots, l_{h-1}\}$. Throughout this paper, we will assume a MRF framework while noting that our results are equally applicable for an CRF.

An MRF defines a neighbourhood relationship (denoted by $\mathcal{E}$) over the random variables such that $(a, b) \in \mathcal{E}$ if, and only if, $v_a$ and $v_b$ are neighbouring random variables. Given an MRF, a *labelling* refers to a function $f$ such that $f : \{0, \cdots, n-1\} \longrightarrow \{0, \cdots, h-1\}$. In other words, the function $f$ assigns to each random variable $v_a \in \mathbf{v}$, a label $l_{f(a)} \in \mathbf{l}$. The probability of the labelling is given by the following Gibbs distribution: $\Pr(f, \mathbf{D}|\boldsymbol{\theta}) = \exp(-Q(f, \mathbf{D}; \boldsymbol{\theta}))/Z(\boldsymbol{\theta})$, where $\boldsymbol{\theta}$ is the parameter of the MRF and $Z(\boldsymbol{\theta})$ is the normalization constant (i.e. the partition function). Assuming a pairwise MRF, the Gibbs energy is given by:

$$Q(f, \mathbf{D}; \boldsymbol{\theta}) = \sum_{v_a \in \mathbf{v}} \theta^1_{a;f(a)} + \sum_{(a,b) \in \mathcal{E}} \theta^2_{ab;f(a)f(b)}, \tag{1}$$

where $\theta^1_{a;f(a)}$ and $\theta^2_{ab;f(a)f(b)}$ are the unary and pairwise potentials respectively. The superscripts '1' and '2' indicate that the unary potential depends on the labelling of one random variable at a time, while the pairwise potential depends on the labelling of two neighbouring random variables.

Clearly, the labelling $f$ which maximizes the posterior $\Pr(f, \mathbf{D}|\boldsymbol{\theta})$ can be obtained by minimizing the Gibbs energy. The problem of obtaining such a labelling $f$ is known as maximum *a posteriori*

(MAP) estimation. In this paper, we consider the problem of MAP estimation of random fields where the pairwise potentials are defined by *truncated convex models* [4]. Formally speaking, the pairwise potentials are of the form

$$\theta^2_{ab;f(a)f(b)} = w_{ab} \min\{d(f(a) - f(b)), M\} \tag{2}$$

where $w_{ab} \geq 0$ for all $(a, b) \in \mathcal{E}$, $d(\cdot)$ is a convex function and $M > 0$ is the truncation factor. Recall that, by the definition of Ishikawa [9], a function $d(\cdot)$ defined at discrete points (specifically over integers) is convex if, and only if, $d(x+1) - 2d(x) + d(x-1) \geq 0$, for all $x \in \mathbb{Z}$. It is assumed that $d(x) = d(-x)$. Otherwise, it can be replaced by $(d(x) + d(-x))/2$ without changing the Gibbs energy of any of the possible labellings of the random field [23]. Examples of pairwise potentials of this form include the truncated linear metric and the truncated quadratic semi-metric, i.e.

$$\theta^2_{ab;f(a)f(b)} = w_{ab} \min\{|f(a) - f(b)|, M\}, \theta^2_{ab;f(a)f(b)} = w_{ab} \min\{(f(a) - f(b))^2, M\}. \tag{3}$$

Before proceeding further, we would like to note here that the method presented in this paper can be trivially extended to *truncated submodular models* (a generalization of truncated convex models). However, we will restrict our discussion to truncated convex models for two reasons: (i) it makes the analysis of our approach easier; and (ii) truncated convex pairwise potentials are commonly used in several problems such as stereo reconstruction, image denoising and inpainting [22]. Note that in the absence of a truncation factor (i.e. when we only have convex pairwise potentials) the exact MAP estimation can be obtained efficiently using the methods of Ishikawa [9] or Veksler [23]. However, minimizing the Gibbs energy in the presence of a truncation factor is well-known to be NP-hard. Given their widespread use, it is not surprising that several approximate MAP estimation algorithms have been proposed in the literature for the truncated convex model. Below, we review such algorithms.

## 1.1 Related Work
Given a random field with truncated convex pairwise potentials, Felzenszwalb and Huttenlocher [6] improved the efficiency of the popular max-product belief propagation (BP) algorithm [19] to obtain the MAP estimate. BP provides the exact MAP estimate when the neighbourhood structure $\mathcal{E}$ of the MRF defines a tree (i.e. it contains no loops). However, for a general MRF, BP provides no bounds on the quality of the approximate MAP labelling obtained. In fact, it is not even guaranteed to converge.

The results of [6] can be used directly to speed-up the tree-reweighted message passing algorithm (TRW) [24] and its sequential variant TRW-S [10]. Both TRW and TRW-S attempt to optimize the Lagrangian dual of the standard linear programming (LP) relaxation of the MAP estimation problem [5, 15, 21, 24]. Unlike BP and TRW, TRW-S is guaranteed to converge. However, it is well-known that TRW-S and other related algorithms [7, 13, 25] suffer from the following problems: (i) they are slower than algorithms based on efficient graph-cuts [22]; and (ii) they only provide a dual solution [10]. The primal solution (i.e. the labelling $f$) is often obtained from the dual solution in an unprincipled manner[1]. Furthermore, it was also observed that, unlike graph-cuts based approaches, TRW-S does not work well when the random field models long range interactions (i.e. when the neighbourhood relationship $\mathcal{E}$ is highly connected) [11]. However, due to the lack of experimental results, it is not clear whether this observation applies to the methods described in [7, 13, 25].

Another way of solving the LP relaxation is to resort to interior point algorithms [3]. Although interior point algorithms are much slower in practice than TRW-S, they have the advantage of providing the primal (possibly fractional) solution of the LP relaxation. Chekuri *et al.* [5] showed that when using certain randomized rounding schemes on the primal solution (to get the final labelling $f$), the following guarantees hold true: (i) for Potts model (i.e. $d(f(a) - f(b)) = |f(a) - f(b)|$ and $M = 1$), we obtain a multiplicative bound[2] of 2; (ii) for the truncated linear metric (i.e.

$$E\left(\frac{Q(f, \mathbf{D}; \boldsymbol{\theta})}{Q(f^*, \mathbf{D}; \boldsymbol{\theta})}\right) \leq \sigma,$$

where $E(\cdot)$ denotes the expectation of its argument under the rounding scheme.

| |
|---|
| **Initialization** |
| - Initialize the labelling to some function $f_1$. For example, $f_1(a) = 0$ for all $v_a \in \mathbf{v}$. |
| **Iteration** |
| - Choose an interval $I_m = [i_m + 1, j_m]$ where $(j_m - i_m) = L$ such that $d(L) \geq M$. |
| - Move from current labelling $f_m$ to a new labelling $f_{m+1}$ such that |
| $\qquad\qquad f_{m+1}(a) = f_m(a)$ or $f_{m+1}(a) \in I_m, \forall v_a \in \mathbf{v}$. |
|    The new labelling is obtained by solving the st-MINCUT problem on a graph described in § 2.1. |
| **Termination** |
| - Stop when there is no further decrease in the Gibbs energy for any interval $I_m$. |

Table 1: *Our Algorithm. As is typical with move making methods, our approach iteratively goes from one labelling to the next by solving an st-*MINCUT *problem. It converges when there remain no moves which reduce the Gibbs energy further.*

$d(f(a) - f(b)) = |f(a) - f(b)|$ and a general $M > 0$), we obtain a multiplicative bound of $2 + \sqrt{2}$; and (iii) for the truncated quadratic semi-metric (i.e. $d(f(a) - f(b)) = (f(a) - f(b))^2$ and a general $M > 0$), we obtain a multiplicative bound of $O(\sqrt{M})$.

The algorithms most related to our approach are the so-called move making methods which rely on solving a series of graph-cut (specifically st-MINCUT) problems. Move making algorithms start with an initial labelling $f_0$ and iteratively minimize the Gibbs energy by moving to a better labelling. At each iteration, (a subset of) random variables have the option of either retaining their old label or taking a new label from a subset of the labels $\mathbf{l}$. For example, in the $\alpha\beta$-swap algorithm [4] the variables currently labelled $l_\alpha$ or $l_\beta$ can either retain their labels or swap them (i.e. some variables labelled $l_\alpha$ can be relabelled as $l_\beta$ and vice versa). The recently proposed range move algorithm [23] modifies this approach such that any variable currently labelled $l_i$ where $i \in [\alpha, \beta]$ can be assigned any label $l_j$ where $j \in [\alpha, \beta]$. Note that the new label $l_j$ can be different from the old label $l_i$, i.e. $i \neq j$. Both these algorithms (i.e. $\alpha\beta$-swap and range move) do not provide any guarantees on the quality of the solution.

In contrast, the $\alpha$-expansion algorithm [4] (where each variable can either retain its label or get assigned the label $l_\alpha$ at an iteration) provides a multiplicative bound of $2$ for the Potts model and $2M$ for the truncated linear metric. Gupta and Tardos [8] generalized the $\alpha$-expansion algorithm for the truncated linear metric and obtained a multiplicative bound of $4$. Komodakis and Tziritas [14] designed a primal-dual algorithm which provides a bound of $2M$ for the truncated quadratic semi-metric. Note that these bounds are inferior to the bounds obtained by the LP relaxation. However, all the above move making algorithms use only a single st-MINCUT at each iteration and are hence, much faster than interior point algorithms, TRW, TRW-S and BP.

### 1.2 Our Results

We further extend the approach of Gupta and Tardos [8] in two ways (section 2). The first extension allows us to handle any truncated convex model (and not just truncated linear). The second extension allows us to consider a potentially larger subset of labels at each iteration compared to [8]. As will be seen in the subsequent analysis (§2.2), these two extensions allow us to solve the MAP estimation problem efficiently using st-MINCUT whilst obtaining the same guarantees as the LP relaxation [5]. Furthermore, our approach does not suffer from the problems of TRW-S mentioned above. In order to demonstrate its practical use, we provide a favourable comparison of our method with several state of the art MAP estimation algorithms (section 3).

## 2 Description of the Algorithm

Table 1 describes the main steps of our approach. Note that unlike the methods described in [4, 23] we will not be able to obtain the optimal move at each iteration. In other words, if in the $m^{th}$ iteration we move from label $f_m$ to $f_{m+1}$ then it is possible that there exists another labelling $f'_{m+1}$ such that $f'_{m+1}(a) = f_m(a)$ or $f'_{m+1}(a) \in I_m$ for all $v_a \in \mathbf{v}$ and $Q(f'_{m+1}, \mathbf{D}; \boldsymbol{\theta}) < Q(f_{m+1}, \mathbf{D}; \boldsymbol{\theta})$. However, our analysis in the next section shows that we are still able to reduce the Gibbs energy sufficiently at each iteration so as to obtain the guarantees of the LP relaxation.

We now turn our attention to designing a method of moving from labelling $f_m$ to $f_{m+1}$. Our approach relies on constructing a graph such that every st-cut on the graph corresponds to a labelling $f'$ of the random variables which satisfies: $f'(a) = f_m(a)$ or $f'(a) \in I_m$, for all $v_a \in \mathbf{v}$. The new labelling $f_{m+1}$ is obtained in two steps: (i) we obtain a labelling $f'$ which corresponds to the

st-MINCUT on our graph; and (ii) we choose the new labelling $f_{m+1}$ as

$$f_{m+1} = \begin{cases} f' & \text{if} \quad Q(f', \mathbf{D}; \boldsymbol{\theta}) \leq Q(f_m, \mathbf{D}; \boldsymbol{\theta}), \\ f_m & \text{otherwise.} \end{cases} \tag{4}$$

Below, we provide the details of the graph construction.

## 2.1 Graph Construction

At each iteration of our algorithm, we are given an interval $I_m = [i_m + 1, j_m]$ of $L$ labels (i.e. $(j_m - i_m) = L$) where $d(L) \geq M$. We also have the current labelling $f_m$ for all the random variables. We construct a directed weighted graph (with non-negative weights) $\mathcal{G}_m = \{\mathcal{V}_m, \mathcal{E}_m, c_m(\cdot, \cdot)\}$ such that for each $v_a \in \mathbf{v}$, we define vertices $\{a_{i_m+1}, a_{i_m+2}, \cdots, a_{j_m}\} \in \mathcal{V}_m$. In addition, as is the case with every st-MINCUT problem, there are two additional vertices called terminals which we denote by $s$ (the source) and $t$ (the sink). The edges $e \in \mathcal{E}_m$ with capacity (i.e. weight) $c_m(e)$ are of two types: (i) those that represent the unary potentials of a labelling corresponding to an st-cut in the graph and; (ii) those that represent the pairwise potentials of the labelling.

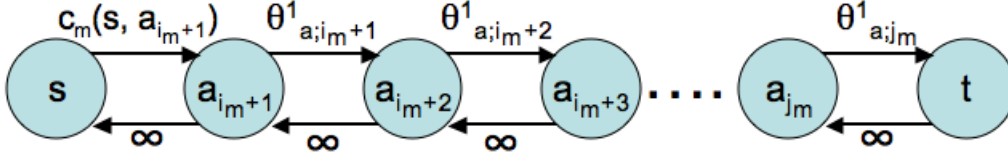

Figure 1: *Part of the graph $\mathcal{G}_m$ containing the terminals and the vertices corresponding to the variable $v_a$. The edges which represent the unary potential of the new labelling are also shown.*

**Representing Unary Potentials** For all random variables $v_a \in \mathbf{v}$, we define the following edges which belong to the set $\mathcal{E}_m$: (i) For all $k \in [i_m + 1, j_m)$, edges $(a_k, a_{k+1})$ have capacity $c_m(a_k, a_{k+1}) = \theta^1_{a;k}$; (ii) For all $k \in [i_m + 1, j_m)$, edges $(a_{k+1}, a_k)$ have capacity $c_m(a_{k+1}, a_k) = \infty$; (iii) Edges $(a_{j_m}, t)$ have capacity $c_m(a_{j_m}, t) = \theta^1_{a;j_m}$; (iv) Edges $(t, a_{j_m})$ have capacity $c_m(t, a_{j_m}) = \infty$; (v) Edges $(s, a_{i_m+1})$ have capacity $c_m(s, a_{i_m+1}) = \theta^1_{a;f_m(a)}$ if $f_m(a) \notin I_m$ and $\infty$ otherwise; and (vi) Edges $(a_{i_m+1}, s)$ have capacity $c_m(a_{i_m+1}, s) = \infty$.

Fig. 1 shows the above edges together with their capacities for one random variable $v_a$. Note that there are two types of edges in the above set: (i) with finite capacity; and (ii) with infinite capacity. Any st-cut with finite cost[3] contains only one of the finite capacity edges for each random variable $v_a$. This is because if an st-cut included more than one finite capacity edge, then by construction it must include at least one infinite capacity edge thereby making its cost infinite [9, 23]. We interpret a finite cost st-cut as a relabelling of the random variables as follows:

$$f'(a) = \begin{cases} k & \text{if st-cut includes edge } (a_k, a_{k+1}) \text{ where } k \in [i_m + 1, j_m), \\ j_m & \text{if st-cut includes edge } (a_{j_m}, t), \\ f_m(a) & \text{if st-cut includes edge } (s, a_{i_m+1}). \end{cases} \tag{5}$$

Note that the sum of the unary potentials for the labelling $f'$ is exactly equal to the cost of the st-cut over the edges defined above. However, the Gibbs energy of the labelling also includes the sum of the pairwise potentials (as shown in equation (1)). Unlike the unary potentials we will not be able to model the sum of pairwise potentials exactly. However, we will be able to obtain its upper bound using the cost of the st-cut over the following edges.

**Representing Pairwise Potentials** For all neighbouring random variables $v_a$ and $v_b$, i.e. $(a, b) \in \mathcal{E}$, we define edges $(a_k, b_{k'}) \in \mathcal{E}_m$ where either one or both of $k$ and $k'$ belong to the set $(i_m + 1, j_m]$ (i.e. at least one of them is different from $i_m + 1$). The capacity of these edges is given by

$$c_m(a_k, b_{k'}) = \frac{w_{ab}}{2} \left( d(k - k' + 1) - 2d(k - k') + d(k - k' - 1) \right). \tag{6}$$

The above capacity is non-negative due to the fact that $w_{ab} \geq 0$ and $d(\cdot)$ is convex. Furthermore, we also add the following edges:

$$c_m(a_k, a_{k+1}) = \frac{w_{ab}}{2} \left( d(L - k + i_m) + d(k - i_m) \right), \forall (a, b) \in \mathcal{E}, k \in [i_m + 1, j_m)$$
$$c_m(b_{k'}, b_{k'+1}) = \frac{w_{ab}}{2} \left( d(L - k' + i_m) + d(k' - i_m) \right), \forall (a, b) \in \mathcal{E}, k' \in [i_m + 1, j_m)$$
$$c_m(a_{j_m}, t) = c_m(b_{j_m}, t) = \frac{w_{ab}}{2} d(L), \forall (a, b) \in \mathcal{E}. \tag{7}$$

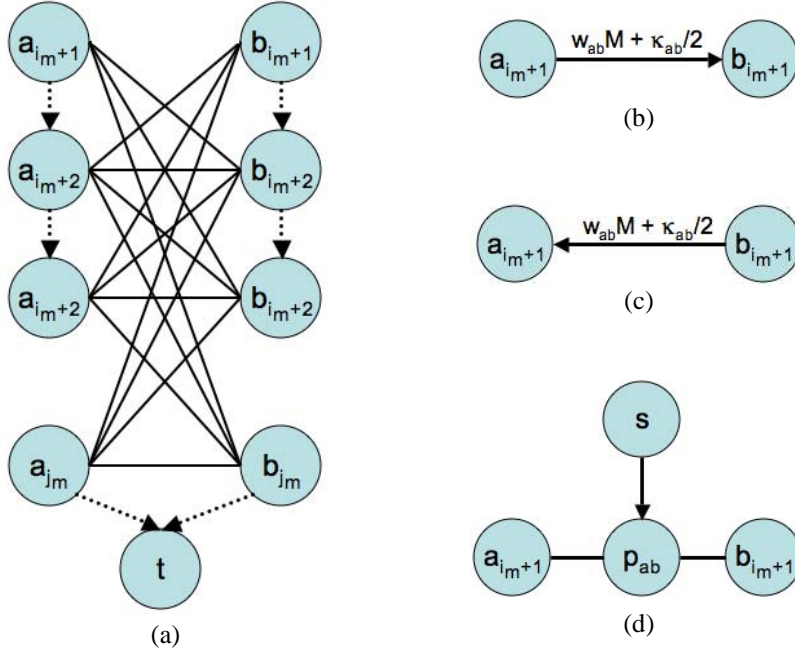

Figure 2: **(a)** *Edges that are used to represent the pairwise potentials of two neighbouring random variables $v_a$ and $v_b$ are shown. Undirected edges indicate that there are opposing edges in both directions with equal capacity (as given by equation 6). Directed dashed edges, with capacities shown in equation (7), are added to ensure that the graph models the convex pairwise potentials correctly.* **(b)** *An additional edge is added when $f_m(a) \in I_m$ and $f_m(b) \notin I_m$. The term $\kappa_{ab} = w_{ab}d(L)$.* **(c)** *A similar additional edge is added when $f_m(a) \notin I_m$ and $f_m(b) \in I_m$.* **(d)** *Five edges, with capacities as shown in equation (8), are added when $f_m(a) \notin I_m$ and $f_m(b) \notin I_m$. Undirected edges indicate the presence of opposing edges with equal capacity.*

Note that in [23] the graph obtained by the edges in equations (6) and (7) was used to find the exact MAP estimate for convex pairwise potentials. A proof that the above edges exactly model convex pairwise potentials up to an additive constant $\kappa_{ab} = w_{ab}d(L)$ can be found in [17]. However, we are concerned with the NP-hard case where the pairwise potentials are truncated. In order to model this case, we incorporate some additional edges to the above set. These additional edges are best described by considering the following three cases for all $(a, b) \in \mathcal{E}$.

- If $f_m(a) \in I_m$ and $f_m(b) \in I_m$ then we do not add any more edges in the graph (see Fig. 2(a)).

- If $f_m(a) \in I_m$ and $f_m(b) \notin I_m$ then we add an edge $(a_{i_m+1}, b_{i_m+1})$ with capacity $w_{ab}M + \kappa_{ab}/2$, where $\kappa_{ab} = w_{ab}d(L)$ is a constant for a given pair of neighbouring random variables $(a, b) \in \mathcal{E}$ (see Fig. 2(b)). Similarly, if $f_m(a) \notin I_m$ and $f_m(b) \in I_m$ then we add an edge $(b_{i_m+1}, a_{i_m+1})$ with capacity $w_{ab}M + \kappa_{ab}/2$ (see Fig. 2(c)).

- If $f_m(a) \notin I_m$ and $f_m(b) \notin I_m$, we introduce a new vertex $p_{ab}$. Using this vertex $p_{ab}$, five edges are defined with the following capacities (see Fig. 2(d)):

$$c_m(a_{i_m+1}, p_{ab}) = c_m(p_{ab}, a_{i_m+1}) = c_m(b_{i_m+1}, p_{ab}) = c_m(p_{ab}, b_{i_m+1}) = w_{ab}M + \kappa_{ab}/2,$$
$$c_m(s, p_{ab}) = \theta^2_{ab;f_m(a),f_m(b)} + \kappa_{ab}. \tag{8}$$

This completes our graph construction. Given the graph $\mathcal{G}_m$ we solve the st-MINCUT problem which provides us with a labelling $f'$ as described in equation (5). The new labelling $f_{m+1}$ is obtained using equation (4). Note that our graph construction is similar to that of Gupta and Tardos [8] with two notable exceptions: (i) we can handle any general truncated convex model and not just truncated linear as in the case of [8]. This is achieved in part by using the graph construction of [23]; and (ii) we have the freedom to choose the value of $L$, while [8] fixed this value to $M$. A logical choice would be to use that value of $L$ which minimizes the worst case multiplicative bound for a particular class of problems. The following properties provide such a value of $L$ for both the truncated linear and the truncated quadratic models. Our worst case multiplicative bounds are exactly those achieved by the LP relaxation (see [5]).

## 2.2 Properties of the Algorithm

For the above graph construction, the following properties hold true:

• The cost of the st-MINCUT provides an upper bound on the Gibbs energy of the labelling $f'$ and hence, on the Gibbs energy of $f_{m+1}$ (see section 2.2 of [17]).

• For the truncated linear metric, our algorithm obtains a multiplicative bound of $2 + \sqrt{2}$ using $L = \sqrt{2}M$ (see section 3, Theorem 1, of [17]). Note that this bound is better than those obtained by $\alpha$-expansion [4] (i.e. $2M$) and its generalization [8] (i.e. $4$).

• For the truncated quadratic semi-metric, our algorithm obtains a multiplicative bound of $O(\sqrt{M})$ using $L = \sqrt{M}$ (see section 3, Theorem 2, of [17]). Note that both $\alpha$-expansion and the approach of Gupta and Tardos provide no bounds for the above case. The primal-dual method of [14] obtains a bound of $2M$ which is clearly inferior to our guarantees.

## 3 Experiments

We tested our approach using both synthetic and standard real data. Below, we describe the experimental setup and the results obtained in detail.

### 3.1 Synthetic Data

**Experimental Setup** We used 100 random fields for both the truncated linear and truncated quadratic models. The variables $\mathbf{v}$ and neighbourhood relationship $\mathcal{E}$ of the random fields described a 4-connected grid graph of size $50 \times 50$. Note that 4-connected grid graphs are widely used to model several problems in Computer Vision [22]. Each variable was allowed to take one of 20 possible labels, i.e. $\mathbf{l} = \{l_0, l_1, \cdots, l_{19}\}$. The parameters of the random field were generated randomly. Specifically, the unary potentials $\theta^1_{a;i}$ were sampled uniformly from the interval $[0, 10]$ while the weights $w_{ab}$, which determine the pairwise potentials, were sampled uniformly from $[0, 5]$. The parameter $M$ was also chosen randomly while taking care that $d(5) \leq M \leq d(10)$.

**Results** Fig. 3 shows the results obtained by our approach and five other state of the art algorithms: $\alpha\beta$-swap, $\alpha$-expansion, BP, TRW-S and the range move algorithm of [23]. We used publicly available code for all previously proposed approaches with the exception of the range move algorithm[4]. As can be seen from the figure, the most accurate approach is the method proposed in this paper, followed closely by the range move algorithm. Recall that, unlike range move, our algorithm is guaranteed to provide the same worst case multiplicative bounds as the LP relaxation. As expected, both the range move algorithm and our method are slower than $\alpha\beta$-swap and $\alpha$-expansion (since each iteration computes an st-MINCUT on a larger graph). However, they are faster than TRW-S, which attempts to minimize the LP relaxation, and BP. We note here that our implementation does not use any clever tricks to speed up the max-flow algorithm (such as those described in [1]) which can potentially decrease the running time by orders of magnitude.

### 3.2 Real Data - Stereo Reconstruction

Given two *epipolar rectified* images $\mathbf{D}_1$ and $\mathbf{D}_2$ of the same scene, the problem of stereo reconstruction is to obtain a correspondence between the pixels of the images. This problem can be modelled using a random field whose variables correspond to pixels of one image (say $\mathbf{D}_1$) and take labels from a set of *disparities* $\mathbf{l} = \{0, 1, \cdots, h-1\}$. A disparity value $i$ for a random variable $a$ denoting pixel $(x, y)$ in $\mathbf{D}_1$ indicates that its corresponding pixel lies in $(x + i, y)$ in the second image.

For the above random field formulation, the unary potentials were defined as in [22] and were truncated at 15. As is typically the case, we chose the neighbourhood relationship $\mathcal{E}$ to define a 4-neighbourhood grid graph. The number of disparities $h$ was set to 20. We experimented using the following truncated convex potentials:

$$\theta^2_{ab;ij} = 50 \min\{|i-j|, 10\}, \theta^2_{ab;ij} = 50 \min\{(i-j)^2, 100\}. \tag{9}$$

The above form of pairwise potentials encourage neighbouring pixels to take similar disparity values which corresponds to our expectations of finding smooth surfaces in natural images. Truncation of pairwise potentials is essential to avoid oversmoothing, as observed in [4, 23]. Note that using spatially varying weights $w_{ab}$ provides better results. However, the main aim of this experiment is to demonstrate the accuracy and speed of our approach and not to design the best possible Gibbs

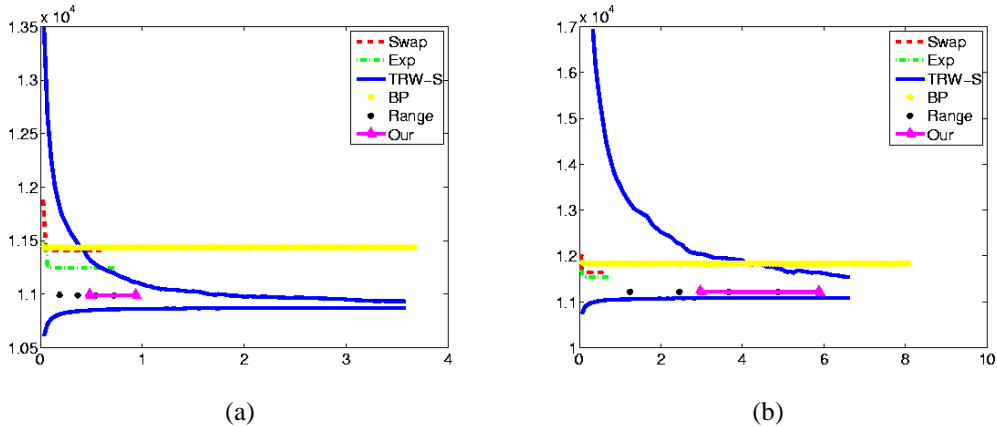

(a)                                                    (b)

Figure 3: *Results of the synthetic experiment.* **(a)** *Truncated linear metric.* **(b)** *Truncated quadratic semi-metric. The x-axis shows the time taken in seconds. The y-axis shows the average Gibbs energy obtained over all 100 random fields using the six algorithms. The lower blue curve is the value of the dual obtained by* TRW-S. *In both the cases, our method and the range move algorithm provide the most accurate solution and are faster than* TRW-S *and* BP.

energy. Table 2 provides the value of the Gibbs energy and the total time taken by all the approaches for a standard stereo pair (Teddy). As in the case of the synthetic experiments, the range move algorithm and our method provide the most accurate solutions while taking less time than TRW-S and BP. Additional experiments on other stereo pairs with similar observations about the performances of the various algorithms can be found in [17]. However, we would again like to emphasize that unlike our method the range move algorithm provides no theoretical guarantees about the quality of the solution.

| Algorithm | Energy-1 | Time-1(s) | Energy-2 | Time-2(s) |
|---|---|---|---|---|
| $\alpha\beta$-swap | 3678200 | 18.48 | 3707268 | 20.25 |
| $\alpha$-expansion | 3677950 | 11.73 | 3687874 | 8.79 |
| TRW-S | 3677578 | 131.65 | 3679563 | 332.94 |
| BP | 3789486 | 272.06 | 5180705 | 331.36 |
| Range Move | 3686844 | 97.23 | **3679552** | 141.78 |
| Our Approach | **3613003** | 120.14 | **3679552** | 191.20 |

Table 2: *The energy obtained and the time taken by the algorithms used in the stereo reconstruction experiment with the Teddy image pair. Columns 2 and 3 : truncated linear metric. Columns 4 and 5: truncated quadratic semi-metric.*

## 4  Discussion

We have presented an st-MINCUT based algorithm for obtaining the approximate MAP estimate of discrete random fields with truncated convex pairwise potentials. Our method improves the multiplicative bound for the truncated linear metric compared to [4, 8] and provides the best known bound for the truncated quadratic semi-metric. Due to the use of only the st-MINCUT problem in its design, it is faster than previous approaches based on the LP relaxation. In fact, its speed can be further improved by a large factor using clever techniques such as those described in [12] (for convex unary potentials) and/or [1] (for general unary potentials). Furthermore, it overcomes the well-known deficiencies of TRW and its variants. Experiments on synthetic and real data problems demonstrate its effectiveness compared to several state of the art algorithms.

The analysis in §2.2 shows that, for the truncated linear and truncated quadratic models, the bound achieved by our move making algorithm over intervals of any length $L$ is equal to that of rounding the LP relaxation's optimal solution using the same intervals [5]. This equivalence also extends to the Potts model (in which case $\alpha$-expansion provides the same bound as the LP relaxation). A natural question would be to ask about the relationship between move making algorithms and the rounding schemes used in convex relaxations. Note that despite recent efforts [14] which analyze certain move making algorithms in the context of primal-dual approaches for the LP relaxation, not many results

are known about their connection with randomized rounding schemes. Although the discussion in §2.2 cannot be trivially generalized to all random fields, it offers a first step towards answering this question. We believe that further exploration in this direction would help improve the understanding of the nature of the MAP estimation problem, e.g. how to derandomize approaches based on convex relaxations. Furthermore, it would also help design efficient move making algorithms for more complex relaxations such as those described in [16].

**Acknowledgments**   The first author was supported by the EU CLASS project and EPSRC grant EP/C006631/1(P). The second author is in receipt of a Royal Society Wolfson Research Merit Award, and would like to acknowledge support from the Royal Society and Wolfson foundation.

## Footnotes

[1] We note here that the recently proposed algorithm in [20] directly provides the primal solution. However, it is much slower than the methods which solve the dual.

[2] Let $f$ be the labelling obtained by an algorithm A (e.g. in this case the LP relaxation followed by the rounding scheme) for a class of MAP estimation problems (e.g. in this case when the pairwise potentials form a Potts model). Let $f^*$ be the optimal labelling. The algorithm A is said to achieve a multiplicative bound of $\sigma$, if for every instance in the class of MAP estimation problems the following holds true:

[3]Recall that the cost of an st-cut is the sum of the capacities of the edges whose starting point lies in the set of vertices containing the source $s$ and whose ending point lies in the set of vertices containing the sink $t$.

[4]When using $\alpha$-expansion with the truncated quadratic semi-metric, all edges with negative capacities in the graph construction were removed, similar to the experiments in [22].

# References

[1]  K. Alahari, P. Kohli, and P. H. S. Torr. Reduce, reuse & recycle: Efficiently solving multi-label MRFs. In *CVPR*, 2008.

[2]  J. Besag. On the statistical analysis of dirty pictures. *Journal of the Royal Statistical Society, Series B*, 48:259–302, 1986.

[3]  S. Boyd and L. Vandenberghe. *Convex Optimization*. Cambridge University Press, 2004.

[4]  Y. Boykov, O. Veksler, and R. Zabih. Fast approximate energy minimization via graph cuts. *PAMI*, 23(11):1222–1239, 2001.

[5]  C. Chekuri, S. Khanna, J. Naor, and L. Zosin. A linear programming formulation and approximation algorithms for the metric labelling problem. *SIAM Journal on Disc. Math.*, 18(3):606–635, 2005.

[6]  P. Felzenszwalb and D. Huttenlocher. Efficient belief propagation for early vision. In *CVPR*, 2004.

[7]  A. Globerson and T. Jaakkola. Fixing max-product: Convergent message passing for MAP LP-relaxations. In *NIPS*, 2007.

[8]  A. Gupta and E. Tardos. A constant factor approximation algorithm for a class of classification problems. In *STOC*, 2000.

[9]  H. Ishikawa. Exact optimization for Markov random fields with convex priors. *PAMI*, 25(10):1333–1336, October 2003.

[10]  V. Kolmogorov. Convergent tree-reweighted message passing for energy minimization. *PAMI*, 28(10):1568–1583, 2006.

[11]  V. Kolmogorov and C. Rother. Comparison of energy minimization algorithms for highly connected graphs. In *ECCV*, pages II: 1–15, 2006.

[12]  V. Kolmogorov and A. Shioura. New algorithms for the dual of the convex cost network flow problem with applications to computer vision. Technical report, University College London, 2007.

[13]  N. Komodakis, N. Paragios, and G. Tziritas. MRF optimization via dual decomposition: Message-passing revisited. In *ICCV*, 2007.

[14]  N. Komodakis and G. Tziritas. Approximate labeling via graph-cuts based on linear programming. *PAMI*, 2007.

[15]  A. Koster, C. van Hoesel, and A. Kolen. The partial constraint satisfaction problem: Facets and lifting theorems. *Operations Research Letters*, 23(3-5):89–97, 1998.

[16]  M. P. Kumar, V. Kolmogorov, and P. H. S. Torr. An analysis of convex relaxations for MAP estimation. In *NIPS*, 2007.

[17]  M. P. Kumar and P. H. S. Torr. Improved moves for truncated convex models. Technical report, University of Oxford, 2008.

[18]  J. Lafferty, A. McCallum, and F. Pereira. Conditional random fields: Probabilistic models for segmenting and labelling sequence data. In *ICML*, 2001.

[19]  J. Pearl. *Probabilistic Reasoning in Intelligent Systems: Networks of Plausible Inference*. Morgan Kauffman, 1988.

[20]  P. Ravikumar, A. Agarwal, and M. Wainwright. Message-passing for graph-structured linear programs: Proximal projections, convergence and rounding schemes. In *ICML*, 2008.

[21]  M. Schlesinger. Sintaksicheskiy analiz dvumernykh zritelnikh singnalov v usloviyakh pomekh (syntactic analysis of two-dimensional visual signals in noisy conditions). *Kibernetika*, 4:113–130, 1976.

[22]  R. Szeliski, R. Zabih, D. Scharstein, O. Veksler, V. Kolmogorov, A. Agarwala, M. Tappen, and C. Rother. A comparative study of energy minimization methods for Markov random fields with smoothness-based priors. *PAMI*, 2008.

[23]  O. Veksler. Graph cut based optimization for MRFs with truncated convex priors. In *CVPR*, 2007.

[24]  M. Wainwright, T. Jaakkola, and A. Willsky. MAP estimation via agreement on trees: Message passing and linear programming. *IEEE Trans. on Information Theory*, 51(11):3697–3717, 2005.

[25]  Y. Weiss, C. Yanover, and T. Meltzer. MAP estimation, linear programming and belief propagation with convex free energies. In *UAI*, 2007.

